# Learning Bounds for Importance Weighting

**Corinna Cortes**
Google Research
New York, NY 10011
corinna@google.com

**Yishay Mansour**
Tel-Aviv University
Tel-Aviv 69978, Israel
mansour@tau.ac.il

**Mehryar Mohri**
Courant Institute and Google
New York, NY 10012
mohri@cims.nyu.edu

## Abstract

This paper presents an analysis of importance weighting for learning from finite samples and gives a series of theoretical and algorithmic results. We point out simple cases where importance weighting can fail, which suggests the need for an analysis of the properties of this technique. We then give both upper and lower bounds for generalization with bounded importance weights and, more significantly, give learning guarantees for the more common case of unbounded importance weights under the weak assumption that the second moment is bounded, a condition related to the Rényi divergence of the training and test distributions. These results are based on a series of novel and general bounds we derive for unbounded loss functions, which are of independent interest. We use these bounds to guide the definition of an alternative reweighting algorithm and report the results of experiments demonstrating its benefits. Finally, we analyze the properties of normalized importance weights which are also commonly used.

## 1   Introduction

In real-world applications of machine learning, often the sampling of the training and test instances may differ, which results in a mismatch between the two distributions. For example, in web search applications, there may be data regarding users who clicked on some advertisement link but little or no information about other users. Similarly, in credit default analyses, there is typically some information available about the credit defaults of customers who were granted credit, but no such information is at hand about rejected costumers. In other problems such as adaptation, the training data available is drawn from a source domain different from the target domain. These issues of biased sampling or adaptation have been long recognized and studied in the statistics literature. There is also a large body of literature dealing with different techniques for sample bias correction [11, 29, 16, 8, 25, 6] or domain adaptation [3, 7, 19, 10, 17] in the recent machine learning and natural language processing literature.

A common technique used in several of these publications for correcting the bias or discrepancy is based on the so-called *importance weighting* technique. This consists of weighting the cost of errors on training instances to emphasize the error on some or de-emphasize it on others, with the objective of correcting the mismatch between the distributions of training and test points, as in sample bias correction, adaptation, and other related contexts such as active learning [24, 14, 8, 19, 5]. Different definitions have been adopted for these weights. A common definition of the weight for point $x$ is $w(x) = P(x)/Q(x)$ where $P$ is the target or test distribution and $Q$ is the distribution according to which training points are drawn. A favorable property of this definition, which is not hard to verify, is that it leads to unbiased estimates of the generalization error [8].

This paper presents an analysis of importance weighting for learning from finite samples. Our study was originally motivated by the observation that, while this corrective technique seems natural, in some cases in practice it does not succeed. An example in dimension two is illustrated by Figure 1. The target distribution $P$ is the even mixture of two Gaussians centered at $(0,0)$ and $(0,2)$ both with

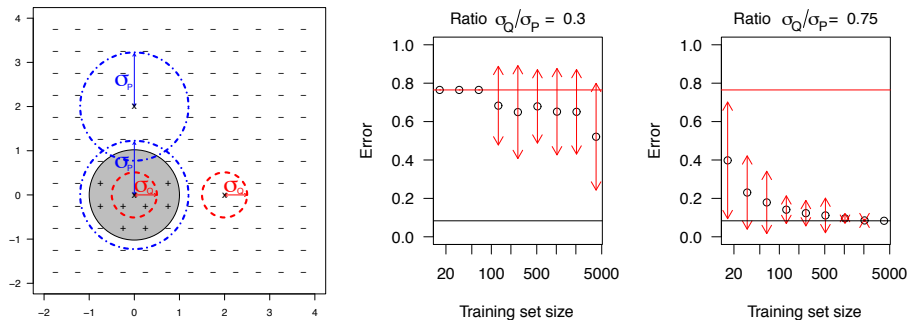

Figure 1: Example of importance weighting. *Left figure*: $P$ (in blue) and $Q$ (in red) are even mixtures of Gaussians. The labels are positive within the unit sphere centered at the origin (in grey), negative elsewhere. The hypothesis class is that of hyperplanes tangent to the unit sphere. *Right figures*: plots of test error vs training sample size using importance weighting for two different values of the ratio $\sigma_Q/\sigma_P$. The results indicate mean values of the error over 40 runs $\pm$ one standard deviation.

standard deviation $\sigma_P$, while the source distribution $Q$ is the even mixture of two Gaussians centered at $(0,0)$ and $(2,0)$ but with standard deviation $\sigma_Q$. The hypothesis class is that of hyperplanes tangent to the unit sphere. The best classifier is selected by empirical risk minimization. As shown in Figure 1, for $\sigma_P/\sigma_Q = .3$, the error of the hypothesis learned using importance weighting is close to $50\%$ even for a training sample of $5{,}000$ points and the standard deviation of the error is quite high. In contrast, for $\sigma_P/\sigma_Q = .75$, convergence occurs relatively rapidly and learning is successful. In Section 4, we discuss other examples where importance weighting does not succeed.

The problem just described is not limited to isolated examples. Similar observations have been made in the past in both the statistics and learning literature, more recently in the context of the analysis of boosting by [9] who suggest that importance weighting must be used with care and highlight the need for convergence bounds and learning guarantees for this technique.

We study the theoretical properties of importance weighting. We show using standard generalization bounds that importance weighting can succeed when the weights are bounded. However, this condition often does not hold in practice. We also show that, remarkably, convergence guarantees can be given even for unbounded weights under the weak assumption that the second moment of the weights is bounded, a condition that relates to the Rényi divergence of $P$ and $Q$. We further extend these bounds to guarantees for other possible reweightings. These results suggest minimizing a bias-variance tradeoff that we discuss and that leads to several algorithmic ideas. We explore in detail an algorithm based on these ideas and report the results of experiments demonstrating its benefits.

Throughout this paper, we consider the case where the weight function $w$ is known. When it is not, it is typically estimated from finite samples. The effect of this estimation error is specifically analyzed by [8]. This setting is closely related to the problem of importance sampling in statistics which is that of estimating the expectation of a random variable according to $P$ while using a sample drawn according to $Q$, with $w$ given [18]. Here, we are concerned with the effect of the weights on learning from finite samples. A different setting is when further full access to $Q$ is assumed, von Neumann's rejection sampling technique [28] can then be used. We note however that it requires $w$ to be bounded by some constant $M$, which is often not guaranteed and is the simplest case of our bounds. Even then, the method is wasteful as it requires on average $M$ samples to obtain one point.

The remainder of this paper is structured as follows. Section 2 introduces the definition of the Rényi divergences and gives some basic properties of the importance weights. In Section 3, we give generalization bounds for importance weighting in the bounded case. We also present a general lower bound indicating the key role played by the Rényi divergence of $P$ and $Q$ in this context. Section 4 deals with the more frequent case of unbounded $w$. Standard generalization bounds do not apply here since the loss function is unbounded. We give novel generalization bounds for unbounded loss functions under the assumption that the second moment is bounded (see Appendix) and use them to derive learning guarantees for importance weighting in this more general setting. In Section 5, we discuss an algorithm inspired by these guarantees for which we report preliminary experimental results. We also discuss why the commonly used remedy of truncating or capping importance weights may not always provide the desired effect of improved performance. Finally, in Section 6, we study

the properties of an alternative reweighting also commonly used which is based on normalized importance weights, and discuss its relationship with the (unnormalized) weights $w$.

## 2 Preliminaries

Let $X$ denote the input space, $Y$ the label set, and let $L\colon Y{\times}Y \to [0,1]$ be a loss function. We denote by $P$ the target distribution and by $Q$ the source distribution according to which training points are drawn. We also denote by $H$ the hypothesis set used by the learning algorithm and by $f\colon X \to Y$ the target labeling function.

### 2.1 Rényi divergences

Our analysis makes use of the notion of Rényi divergence, an information theoretical measure of the difference between two distributions directly relevant to the study of importance weighting. For $\alpha \geq 0$, the Rényi divergence $D_\alpha(P\|Q)$ between distributions $P$ and $Q$ is defined by [23]

$$D_\alpha(P\|Q) = \frac{1}{\alpha - 1} \log_2 \sum_x P(x) \left( \frac{P(x)}{Q(x)} \right)^{\alpha-1}. \tag{1}$$

The Rényi divergence is a non-negative quantity and for any $\alpha > 0$, $D_\alpha(P\|Q) = 0$ iff $P = Q$. For $\alpha = 1$, it coincides with the relative entropy. We denote by $d_\alpha(P\|Q)$ the exponential in base 2 of the Rényi divergence $D_\alpha(P\|Q)$:

$$d_\alpha(P\|Q) = 2^{D_\alpha(P\|Q)} = \left[ \sum_x \frac{P^\alpha(x)}{Q^{\alpha-1}(x)} \right]^{\frac{1}{\alpha-1}}. \tag{2}$$

### 2.2 Importance weights

The *importance weight* for distributions $P$ and $Q$ is defined by $w(x) = P(x)/Q(x)$. In the following, the expectations are taken with respect to $Q$.

**Lemma 1.** *The following identities hold for the expectation, second moment, and variance of $w$:*

$$\mathrm{E}[w] = 1 \qquad \mathrm{E}[w^2] = d_2(P\|Q) \qquad \sigma^2(w) = d_2(P\|Q) - 1. \tag{3}$$

*Proof.* The first equality is immediate. The second moment of $w$ can be expressed as follows in terms of the Rényi divergence:

$$\mathop{\mathrm{E}}_{Q}[w^2] = \sum_{x \in X} w^2(x)\, Q(x) = \sum_{x \in X} \left( \frac{P(x)}{Q(x)} \right)^2 Q(x) = \sum_{x \in X} P(x) \left( \frac{P(x)}{Q(x)} \right) = d_2(P\|Q).$$

Thus, the variance of $w$ is given by $\sigma^2(w) = \mathrm{E}_Q[w^2] - \mathrm{E}_Q[w]^2 = d_2(P\|Q) - 1$. $\qquad\square$

For any hypothesis $h \in H$, we denote by $R(h)$ its *loss* and by $\widehat{R}_w(h)$ its *weighted empirical loss*:

$$R(h) = \mathop{\mathrm{E}}_{x \sim P}[L(h(x), f(x))] \qquad \widehat{R}_w(h) = \frac{1}{m} \sum_{i=1}^{m} w(x_i)\, L(h(x_i), f(x_i)).$$

We shall use the abbreviated notation $L_h(x)$ for $L(h(x), f(x))$, in the absence of any ambiguity about the target function $f$. Note that the unnormalized importance weighting of the loss is unbiased:

$$\mathop{\mathrm{E}}_{Q}[w(x)L_h(x)] = \sum_x \frac{P(x)}{Q(x)} L_h(x)\, Q(x) = \sum_x P(x) L_h(x) = R(h).$$

The following lemma gives a bound on the second moment.

**Lemma 2.** *For all $\alpha > 0$ and $x \in X$, the second moment of the importance weighted loss can be bounded as follows:*

$$\mathop{\mathrm{E}}_{x \sim Q}[w^2(x)\, L_h^2(x)] \leq d_{\alpha+1}(P\|Q)\, R(h)^{1-\frac{1}{\alpha}}. \tag{4}$$

*For $\alpha = 1$, this becomes $R(h)^2 \leq \mathrm{E}_{x \sim Q}[w^2(x)\, L_h^2(x)] \leq d_2(P\|Q)$.*

*Proof.* The second moment can be bounded as follows:

$$\mathop{\mathrm{E}}_{x \sim Q}[w^2(x)\, L_h^2(x)] = \sum_x Q(x)\left[\frac{P(x)}{Q(x)}\right]^2 L_h^2(x) = \sum_x P(x)^{\frac{1}{\alpha}}\left[\frac{P(x)}{Q(x)}\right] P(x)^{\frac{\alpha-1}{\alpha}} L_h^2(x)$$

$$\leq \left[\sum_x P(x)\left[\frac{P(x)}{Q(x)}\right]^\alpha\right]^{\frac{1}{\alpha}}\left[\sum_x P(x)\, L_h^{\frac{2\alpha}{\alpha-1}}(x)\right]^{\frac{\alpha-1}{\alpha}} \quad \text{(Hölder's inequality)}$$

$$= d_{\alpha+1}(P\|Q)\left[\sum_x P(x)\, L_h(x) L_h^{\frac{\alpha+1}{\alpha-1}}(x)\right]^{\frac{\alpha-1}{\alpha}}$$

$$\leq d_{\alpha+1}(P\|Q)\, R(h)^{1-\frac{1}{\alpha}} B^{1+\frac{1}{\alpha}} = d_{\alpha+1}(P\|Q)\, R(h)^{1-\frac{1}{\alpha}}. \qquad \square$$

## 3   Learning Guarantees - Bounded Case

Note that $\sup_x w(x) = \sup_x \frac{P(x)}{Q(x)} = d_\infty(P\|Q)$. We first examine the case $d_\infty(P\|Q) < +\infty$ and use the notation $M = d_\infty(P\|Q)$. The following proposition follows then directly Hoeffding's inequality.

**Proposition 1** (single hypothesis)**.** *Fix* $h \in H$. *For any* $\delta > 0$, *with probability at least* $1 - \delta$,

$$|R(h) - \widehat{R}_w(h)| \leq M\sqrt{\frac{\log(2/\delta)}{2m}}.$$

The upper bound $M$, though finite, can be quite large. The following theorem provides a more favorable bound as a function of the ratio $M/m$ when any of the moments of $w$, $d_{\alpha+1}(P\|Q)$, is finite, which is the case when $d_\infty(P\|Q) < \infty$ since the Rényi divergence is a non-decreasing function of $\alpha$ [23, 2], in particular:

$$\forall \alpha > 0, \quad d_{\alpha+1}(P\|Q) \leq d_\infty(P\|Q). \tag{5}$$

**Theorem 1** (single hypothesis)**.** *Fix* $h \in H$. *Then, for any* $\alpha \geq 1$, *for any* $\delta > 0$, *with probability at least* $1 - \delta$, *the following bound holds for the importance weighting method:*

$$R(h) \leq \widehat{R}_w(h) + \frac{2M\log\frac{1}{\delta}}{3m} + \sqrt{\frac{2\left[d_{\alpha+1}(P\|Q)\, R(h)^{1-\frac{1}{\alpha}} - R(h)^2\right]\log\frac{1}{\delta}}{m}}. \tag{6}$$

*For* $\alpha = 1$ *after further simplification, this gives* $R(h) \leq \widehat{R}_w(h) + \frac{2M\log\frac{1}{\delta}}{3m} + \sqrt{\frac{2d_2(P\|Q)\log\frac{1}{\delta}}{m}}$.

*Proof.* Let $Z$ denote the random variable $w(x)\, L_h(x) - R(h)$. Then, $|Z| \leq M$. By lemma 2, the variance of the random variable $Z$ can be bounded in terms of the Rényi divergence $d_{\alpha+1}(P\|Q)$:

$$\sigma^2(Z) = \mathop{\mathrm{E}}_Q[w^2(x)\, L_h(x)^2] - R(h)^2 \leq d_{\alpha+1}(P\|Q)\, R(h)^{1-\frac{1}{\alpha}} - R(h)^2.$$

Thus, by Bernstein's inequality [4], it follows that:

$$\Pr[R(h) - \widehat{R}_w(h) > \epsilon] \leq \exp\left(\frac{-m\epsilon^2/2}{\sigma^2(Z) + \epsilon M/3}\right).$$

Setting $\delta$ to match this upper bound shows that with probability at least $1 - \delta$, the following bound holds for the importance weighting method:

$$R(h) \leq \widehat{R}_w(h) + \frac{M\log\frac{1}{\delta}}{3m} + \sqrt{\frac{M^2\log^2\frac{1}{\delta}}{9m^2} + \frac{2\sigma^2(Z)\log\frac{1}{\delta}}{m}}.$$

Using the sub-additivity of $\sqrt{\cdot}$ leads to the simpler expression

$$R(h) \leq \widehat{R}_w(h) + \frac{2M\log\frac{1}{\delta}}{3m} + \sqrt{\frac{2\sigma^2(Z)\log\frac{1}{\delta}}{m}}. \qquad \square$$

These results can be straightforwardly extended to general hypothesis sets. In particular, for a finite hypothesis set and for $\alpha = 1$, the application of the union bound yields the following result.

**Theorem 2** (finite hypothesis set). *Let $H$ be a finite hypothesis set. Then, for any $\delta > 0$, with probability at least $1-\delta$, the following bound holds for the importance weighting method:*

$$R(h) \leq \widehat{R}_w(h) + \frac{2M(\log|H| + \log\frac{1}{\delta})}{3m} + \sqrt{\frac{2d_2(P\|Q)(\log|H| + \log\frac{1}{\delta})}{m}}. \tag{7}$$

For infinite hypothesis sets, a similar result can be shown straightforwardly using covering numbers instead of $|H|$ or a related measure based on samples of size $m$ [20].

In the following proposition, we give a lower bound that further emphasizes the role of the Rényi divergence of the second order in the convergence of importance weighting in the bounded case.

**Proposition 2** (Lower bound). *Assume that $M < \infty$ and $\sigma^2(w)/M^2 \geq 1/m$. Assume that $H$ contains a hypothesis $h_0$ such that $L_{h_0}(x) = 1$ for all $x$. Then, there exists an absolute constant $c$, $c = 2/41^2$, such that*

$$\Pr\left[\sup_{h \in H} |R(h) - \widehat{R}_w(h)| \geq \sqrt{\frac{d_2(P\|Q) - 1}{4m}}\right] \geq c > 0. \tag{8}$$

*Proof.* Let $\sigma_H = \sup_{h \in H} \sigma(wL_h)$. If for all $x \in X$, $L_{h_0}(x) = 1$, then $\sigma^2(wL_{h_0}) = d_2(P\|Q) - 1 = \sigma^2(w) = \sigma_H^2$. The result then follows a general theorem, Theorem 9 proven in the Appendix. $\square$

## 4  Learning Guarantees - Unbounded Case

The condition $d_\infty(P\|Q) < \infty$ assumed in the previous section does not always hold, even in some natural cases, as illustrated by the following examples.

### 4.1  Examples

Assume that $P$ and $Q$ both follow a Gaussian distribution with the standard deviations $\sigma_P$ and $\sigma_Q$ and with means $\mu$ and $\mu'$:

$$P(x) = \frac{1}{\sqrt{2\pi}\sigma_P} \exp\left[-\frac{(x-\mu)^2}{2\sigma_P^2}\right] \quad Q(x) = \frac{1}{\sqrt{2\pi}\sigma_Q} \exp\left[-\frac{(x-\mu')^2}{2\sigma_Q^2}\right].$$

In that case, $\frac{P(x)}{Q(x)} = \frac{\sigma_Q}{\sigma_P}\exp\left[-\frac{\sigma_Q^2(x-\mu)^2 - \sigma_P^2(x-\mu')^2}{2\sigma_P^2\sigma_Q^2}\right]$, thus, even for $\sigma_P = \sigma_Q$ and $\mu \neq \mu'$ the importance weights are unbounded, $d_\infty(P\|Q) = \sup_x \frac{P(x)}{Q(x)} = +\infty$, and the bound of Theorem 1 is not informative. The Rényi divergence of the second order is given by:

$$\begin{aligned}
d_2(P\|Q) &= \frac{\sigma_Q}{\sigma_P} \int_{-\infty}^{+\infty} \exp\left[-\frac{\sigma_Q^2(x-\mu)^2 - \sigma_P^2(x-\mu')^2}{2\sigma_P^2\sigma_Q^2}\right] P(x)dx \\
&= \frac{\sigma_Q}{\sigma_P^2\sqrt{2\pi}} \int_{-\infty}^{+\infty} \exp\left[-\frac{2\sigma_Q^2(x-\mu)^2 - \sigma_P^2(x-\mu')^2}{2\sigma_P^2\sigma_Q^2}\right] dx.
\end{aligned}$$

That is, for $\sigma_Q > \frac{\sqrt{2}}{2}\sigma_P$ the variance of the importance weights is bounded. By the additivity property of the Rényi divergence, a similar situation holds for the product and sums of such Gaussian distributions. Hence, in the rightmost example of Figure 1, the importance weights are unbounded, but their second moment is bounded. In the next section we provide learning guarantees even for this setting in agreement with the results observed. For $\sigma_Q = 0.3\sigma_P$, the same favorable guarantees do not hold, and, as illustrated in Figure 1, learning is significantly more difficult.

This example of Gaussians can further illustrate what can go wrong in importance weighting. Assume that $\mu = \mu' = 0$, $\sigma_Q = 1$ and $\sigma_P = 10$. One could have expected this to be an easy case for importance weighting since sampling from $Q$ provides useful information about $P$. The problem is, however, that a sample from $Q$ will contain a very small number of points far from the mean (of either negative or positive label) and that these points will be assigned very large weights. For a sample of size $m$ and $\sigma_Q = 1$, the expected value of an extreme point is $\sqrt{2\log m} - o(1)$ and its

weight will be in the order of $m^{-1/\sigma_P^2 + 1/\sigma_Q^2} = m^{0.99}$. Therefore, a few extreme points will dominate all other weights and necessarily have a huge influence on the selection of a hypothesis by the learning algorithm.

Another related example is when $\sigma_Q = \sigma_P = 1$ and $\mu' = 0$. Let $\mu \gg 0$ depend on the sample size $m$. If $\mu$ is large enough compared to $\log(m)$, then, with high probability, all the weights will be negligible. This is especially problematic, since the estimate of the probability of any event would be negligible (in fact both an event and its complement). If we normalize the weights, the issue is overcome, but then, with high probability, the maximum weight dominates the sum of all other weights, reverting the situation back to that of the previous example.

## 4.2 Importance weighting learning bounds - unbounded case

As in these examples, in practice, the importance weights are typically not bounded. However, we shall show that, remarkably, under the weak assumption that the second moment of the weights $w$, $d_2(P\|Q)$, is bounded, generalization bounds can be given for this case as well. The following result relies on a general learning bound for unbounded loss functions proven in the Appendix (Corollary 1). We denote by $\mathrm{Pdim}(U)$ the pseudo-dimension of a real-valued function class $U$ [21].

**Theorem 3.** *Let $H$ be a hypothesis set such that $\mathrm{Pdim}(\{L_h(x)\colon h \in H\}) = p < \infty$. Assume that $d_2(P\|Q) < +\infty$ and $w(x) \neq 0$ for all $x$. Then, for any $\delta > 0$, with probability at least $1 - \delta$, the following holds:*

$$R(h) \leq \widehat{R}_w(h) + 2^{5/4}\sqrt{d_2(P\|Q)} \sqrt[8]{\frac{p \log \frac{2me}{p} + \log \frac{4}{\delta}}{m}}.$$

*Proof.* Since $d_2(P\|Q) < +\infty$, the second moment of $w(x)L_h(x)$ is finite and upper bounded by $d_2(P\|Q)$ (Lemma 2). Thus, by Corollary 1, we can write

$$\Pr\left[\sup_{h \in H} \frac{R(h) - \widehat{R}_w(h)}{\sqrt{d_2(P\|Q)}} > \epsilon\right] \leq 4 \exp\left(p \log \frac{2em}{p} - \frac{m\epsilon^{8/3}}{4^{5/3}}\right),$$

where $p$ is the pseudo-dimension of the function class $H'' = \{w(x)L_h(x)\colon h \in H\}$. We now show that $p = \mathrm{Pdim}(\{L_h(x)\colon h \in H\})$. Let $H'$ denote $\{L_h(x)\colon h \in H\}$. Let $A = \{x_1, \ldots, x_k\}$ be a set shattered by $H''$. Then, there exist real numbers $r_1, \ldots, r_k$ such that for any subset $B \subseteq A$ there exists $h \in H$ such that

$$\forall x_i \in B, \quad w(x_i)L_h(x_i) \geq r_i \qquad \forall x_i \in A - B, \quad w(x_i)L_h(x_i) < r_i. \qquad (9)$$

Since by assumption $w(x_i) > 0$ for all $i \in [1, k]$, this implies that

$$\forall x_i \in B, \quad L_h(x_i) \geq r_i/w(x_i) \qquad \forall x_i \in A - B, \quad L_h(x_i) < r_i/w(x_i). \qquad (10)$$

Thus, $H'$ shatters $A$ with the witnesses $s_i = r_i/w(x_i), i \in [1, k]$. Using the same observations, it is straightforward to see that conversely, any set shattered by $H'$ is shattered by $H''$. $\qquad \square$

The convergence rate of the bound is slightly weaker ($O(m^{-3/8})$) than in the bounded case ($O(m^{-1/2})$). A faster convergence can be obtained however using the more precise bound of Theorem 8 at the expense of readability. The Rényi divergence $d_2(P\|Q)$ seems to play a critical role in the bound and thus in the convergence of importance weighting in the unbounded case.

## 5 Alternative reweighting algorithms

The previous analysis can be generalized to the case of an arbitrary positive function $u\colon X \to \mathbb{R}$, $u > 0$. Let $\widehat{R}_u(h) = \frac{1}{m}\sum_{i=1}^{m} u(x_i)L_h(x_i)$ and let $\widehat{Q}$ denote the empirical distribution.

**Theorem 4.** *Let $H$ be a hypothesis set such that $\mathrm{Pdim}(\{L_h(x)\colon h \in H\}) = p < \infty$. Assume that $0 < \mathrm{E}_Q[u^2(x)] < +\infty$ and $u(x) \neq 0$ for all $x$. Then, for any $\delta > 0$, with probability at least $1 - \delta$, the following holds:*

$$|R(h) - \widehat{R}_u(h)| \leq \left|\underset{Q}{\mathrm{E}}\left[[w(x) - u(x)]L_h(x)\right]\right| +$$

$$2^{5/4} \max\left(\sqrt{\mathrm{E}_Q[u^2(x)L_h^2(x)]}, \sqrt{\mathrm{E}_{\widehat{Q}}[u^2(x)L_h^2(x)]}\right) \sqrt[8]{\frac{p \log \frac{2me}{p} + \log \frac{4}{\delta}}{m}}.$$

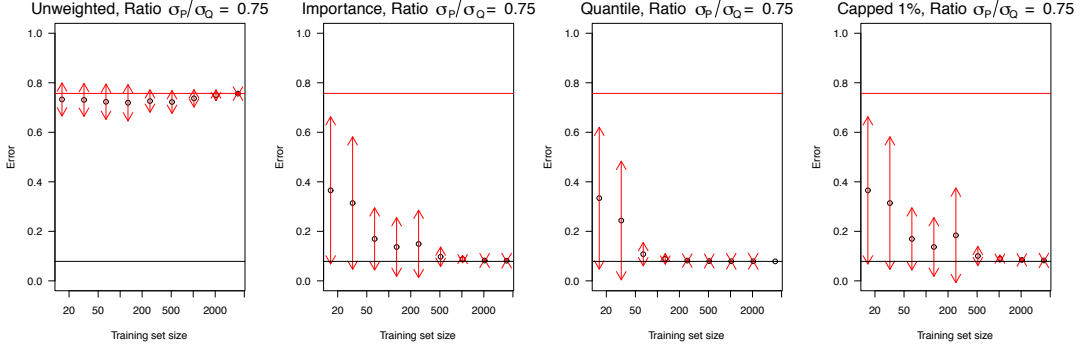

Figure 2: Comparison of the convergence of 4 different algorithms for the learning task of Figure 1: learning with equal weights for all examples (Unweighted), Importance weighting, using Quantiles to parameterize the function $u$, and Capping the largest weights.

*Proof.* Since $R(h) = \mathrm{E}[w(x)L_h(x)]$, we can write

$$R(h) - \widehat{R}_u(h) = \underset{Q}{\mathrm{E}} \left[ [w(x) - u(x)]L_h(x) \right] + \mathrm{E}[u(x)L_h(x)] - \widehat{R}_u(h),$$

and thus

$$\left| R(h) - \widehat{R}_u(h) \right| \leq \left| \underset{Q}{\mathrm{E}} \left[ [w(x) - u(x)]L_h(x) \right] \right| + \left| \mathrm{E}[u(x)L_h(x)] - \widehat{R}_u(h) \right|.$$

By Corollary 2 applied to the function $u L_h$, $\left| \mathrm{E}[u(x)L_h(x)] - \widehat{R}_u(h) \right|$ can be bounded by $2^{5/4} \max(\sqrt{\mathrm{E}_Q[u^2(x)L_h^2(x)]}, \sqrt{\mathrm{E}_{\widehat{Q}}[u^2(x)L_h^2(x)]}) \sqrt[\frac{3}{8}]{\frac{p \log \frac{2me}{p} + \log \frac{4}{\delta}}{m}}$ with probability $1 - \delta$, with $p = \mathrm{Pdim}(\{L_h(x) \colon h \in H\})$ by a proof similar to that of Theorem 3. $\square$

The theorem suggests that other functions $u$ than $w$ can be used to reweight the cost of an error on each training point by minimizing the upper bound, which is a trade-off between the bias term $\left| \mathrm{E}_Q[(w(x)-u(x))L_h(x)] \right|$ and the second moment $\max\left( \sqrt{\mathrm{E}_Q[u^2(x)L_h^2(x)]}, \sqrt{\mathrm{E}_{\widehat{Q}}[u^2(x)L_h^2(x)]} \right)$, where the coefficients are explicitly given. Function $u$ can be selected from different families. Using an upper bound on these quantities that is independent of $h$ and a multiplicative bound of the form

$$\max\left( \sqrt{\underset{Q}{\mathrm{E}}[u^2]}, \sqrt{\underset{\widehat{Q}}{\mathrm{E}}[u^2]} \right) \leq \sqrt{\underset{Q}{\mathrm{E}}[u^2]} \left( 1 + O(1/\sqrt{m}) \right),$$

leads to the following optimization problem:

$$\min_{u \in U} \underset{Q}{\mathrm{E}} \left[ |w(x) - u(x)| \right] + \gamma \sqrt{\underset{Q}{\mathrm{E}}[u^2]}, \tag{11}$$

where $\gamma > 0$ is a parameter controlling the trade-off between bias and variance minimization and where $U$ is a family of possible weight functions out of which $u$ is selected.

Here, we consider a family of functions $U$ parameterized by the quantiles $q$ of the weight function $w$. A function $u_q \in U$ is then defined as follows: within each quantile, the value taken by $u_q$ is the average of $w$ over that quantile. For small values of $\gamma$, the bias term dominates, and very fine-grained quantiles minimize the bound of equation (11). For large values of $\gamma$ the variance term dominates and the bound is minimized by using just one quantile, corresponding to an even weighting of the training examples. Hence by varying $\gamma$ from small to large values, the algorithm interpolates between standard importance weighting with just one example per quantile, and unweighted learning where all examples are given the same weight. Figure 2 also shows the results of experiments for the learning task of Figure 1 using the algorithm defined by (11) with this family of functions. The optimal $q$ is determined by 10-fold cross-validation. We see that a more rapid convergence can be obtained by using these weights compared to the standard importance weights $w$.

Another natural family of functions is that of thresholded versions of the importance weights $\{u_\theta \colon \theta > 0, \forall x \in X, u_\theta(x) = \min(w(x), \theta)\}$. In fact, in practice, users often cap importance weights by choosing an arbitrary value $\theta$. The advantage of this family is that, by definition, the weights are

bounded. However, in some cases, larger weights could be critical to achieve a better performance. Figure 2 illustrates the performance of this approach. Compared to importance weighting, no change in performance is observed until the largest 1% of the weights are capped, in which case we only observe a performance degradation. We expect the thresholding to be less beneficial when the large weights reflect the true $w$ and are not an artifact of estimation uncertainties.

## 6 Relationship between normalized and unnormalized weights

An alternative approach based on the weight function $w = P(x)/Q(x)$ consists of normalizing the weights. Thus, while in the unnormalized case the unweighted empirical error is replaced by

$$\frac{1}{m} \sum_{i=1}^{m} w(x_i) \, L_h(x_i) = \sum_{i=1}^{m} \frac{w(x_i)}{m} \, L_h(x_i),$$

in the normalized case it is replaced by

$$\sum_{i=1}^{m} \frac{w(x_i)}{W} \, L_h(x_i),$$

with $W = \sum_{i=1}^{m} w(x_i)$. We refer to $\widehat{w}(x) = w(x)/W$ as the *normalized importance weight*. An advantage of the normalized weights is that they are by definition bounded by one. However, the price to pay for this benefit is the fact that the weights are no more unbiased. In fact, several issues similar to those we pointed out in the Section 4 affect the normalized weights as well.

Here, we maintain the assumption that the second moment of the importance weights is bounded and analyze the relationship between normalized and unnormalized weights. We show that, under this assumption, normalized and unnormalized weights are in fact very close, with high probability.

Observe that for any $i \in [1, m]$,

$$\widehat{w}(x_i) - \frac{w(x_i)}{m} = w(x_i) \left[ \frac{1}{W} - \frac{1}{m} \right] = \frac{w(x_i)}{W} \left[ 1 - \frac{W}{m} \right].$$

Thus, since $\frac{w(x_i)}{W} \le 1$, we can write $\left| \widehat{w}(x_i) - \frac{w(x_i)}{m} \right| \le \left| 1 - \frac{W}{m} \right|$. Since $\mathrm{E}[w(x)] = 1$, we also have $\mathrm{E}_S[W] = \frac{1}{m} \sum_{k=1}^{m} \mathrm{E}[w(x_k)] = 1$. Thus, by Corollary 2, for any $\delta > 0$, with probability at least $1 - \delta$, the following inequality holds

$$\left| 1 - \frac{W}{m} \right| \le 2^{5/4} \max \left\{ \sqrt{d_2(P\|Q)}, \sqrt{d_2(P\|\widehat{Q})} \right\} \sqrt[\frac{3}{8}]{\frac{\log 2me + \log \frac{4}{\delta}}{m}},$$

which implies the same upper bound on $\left| \widehat{w}(x_i) - \frac{w(x_i)}{m} \right|$, simultaneously for all $i \in [1, m]$.

## 7 Conclusion

We presented a series of theoretical results for importance weighting both in the bounded weights case and in the more general unbounded case under the assumption that the second moment of the weights is bounded. We also initiated a preliminary exploration of alternative weights and showed its benefits. A more systematic study of new algorithms based on these learning guarantees could lead to even more beneficial and practically useful results. Several of the learning guarantees we gave depend on the Rényi divergence of the distributions $P$ and $Q$. Accurately estimating that quantity is thus critical and should motivate further studies of the convergence of its estimates from finite samples. Finally, our novel unbounded loss learning bounds are of independent interest and could be useful in a variety of other contexts.

## References

[1] M. Anthony and J. Shawe-Taylor. A result of Vapnik with applications. *Discrete Applied Mathematics*, 47:207 – 217, 1993.

[2] C. Arndt. *Information Measures: Information and its Description in Science and Engineering*. Signals and Communication Technology. Springer Verlag, 2004.

[3] S. Ben-David, J. Blitzer, K. Crammer, and F. Pereira. Analysis of representations for domain adaptation. *NIPS*, 2007.

[4] S. N. Bernstein. Sur l'extension du théorème limite du calcul des probabilités aux sommes de quantités dépendantes. *Mathematische Annalen*, 97:1–59, 1927.

[5] A. Beygelzimer, S. Dasgupta, and J. Langford. Importance weighted active learning. In *ICML*, pages 49–56, New York, NY, USA, 2009.

[6] S. Bickel, M. Brückner, and T. Scheffer. Discriminative learning for differing training and test distributions. In *ICML*, pages 81–88, 2007.

[7] J. Blitzer, K. Crammer, A. Kulesza, F. Pereira, and J. Wortman. Learning bounds for domain adaptation. *NIPS 2007*, 2008.

[8] C. Cortes, M. Mohri, M. Riley, and A. Rostamizadeh. Sample selection bias correction theory. In *ALT*, 2008.

[9] S. Dasgupta and P. M. Long. Boosting with diverse base classifiers. In *COLT*, 2003.

[10] H. Daumé III and D. Marcu. Domain adaptation for statistical classifiers. *Journal of Artificial Intelligence Research*, 26:101–126, 2006.

[11] M. Dudík, R. E. Schapire, and S. J. Phillips. Correcting sample selection bias in maximum entropy density estimation. In *NIPS*, 2006.

[12] R. M. Dudley. A course on empirical processes. *Lecture Notes in Math*., 1097:2 – 142, 1984.

[13] R. M. Dudley. Universal Donsker classes and metric entropy. *Annals of Probability*, 14(4):1306 – 1326, 1987.

[14] C. Elkan. The foundations of cost-sensitive learning. In *IJCAI*, pages 973–978, 2001.

[15] D. Haussler. Decision theoretic generalizations of the PAC model for neural net and other learning applications. *Inf. Comput*., 100(1):78–150, 1992.

[16] J. Huang, A. J. Smola, A. Gretton, K. M. Borgwardt, and B. Schölkopf. Correcting sample selection bias by unlabeled data. In *NIPS*, volume 19, pages 601–608, 2006.

[17] J. Jiang and C. Zhai. Instance Weighting for Domain Adaptation in NLP. In *ACL*, 2007.

[18] J. S. Liu. *Monte Carlo strategies in scientific computing*. Springer, 2001.

[19] Y. Mansour, M. Mohri, and A. Rostamizadeh. Domain adaptation: Learning bounds and algorithms. In *COLT*, 2009.

[20] A. Maurer and M. Pontil. Empirical bernstein bounds and sample-variance penalization. In *COLT*, Montréal, Canada, June 2009. Omnipress.

[21] D. Pollard. *Convergence of Stochastic Processess*. Springer, New York, 1984.

[22] D. Pollard. Asymptotics via empirical processes. *Statistical Science*, 4(4):341 – 366, 1989.

[23] A. Rényi. On measures of information and entropy. In *Proceedings of the 4th Berkeley Symposium on Mathematics, Statistics and Probability*, page 547561, 1960.

[24] H. Shimodaira. Improving predictive inference under covariate shift by weighting the log-likelihood function. *Journal of Statistical Planning and Inference*, 90(2):227–244, 2000.

[25] M. Sugiyama, S. Nakajima, H. Kashima, P. von Bünau, and M. Kawanabe. Direct importance estimation with model selection and its application to covariate shift adaptation. In *NIPS*, 2008.

[26] V. N. Vapnik. *Statistical Learning Theory*. John Wiley & Sons, 1998.

[27] V. N. Vapnik. *Estimation of Dependences Based on Empirical Data, 2nd ed*. Springer, 2006.

[28] J. von Neumann. Various techniques used in connection with random digits. Monte Carlo methods. *Nat. Bureau Standards*, 12:36–38, 1951.

[29] B. Zadrozny, J. Langford, and N. Abe. Cost-sensitive learning by cost-proportionate example weighting. In *ICDM*, 2003.

